# Algorithms for Infinitely Many-Armed Bandits

**Yizao Wang**[*]
Department of Statistics - University of Michigan
437 West Hall, 1085 South University, Ann Arbor, MI, 48109-1107, USA
yizwang@umich.edu

**Jean-Yves Audibert**
Université Paris Est, Ecole des Ponts, ParisTech, Certis
& Willow - ENS / INRIA, Paris, France
audibert@certis.enpc.fr

**Rémi Munos**
INRIA Lille - Nord Europe, SequeL project,
40 avenue Halley, 59650 Villeneuve d'Ascq, France
remi.munos@inria.fr

## Abstract

We consider multi-armed bandit problems where the number of arms is larger than the possible number of experiments. We make a stochastic assumption on the mean-reward of a new selected arm which characterizes its probability of being a near-optimal arm. Our assumption is weaker than in previous works. We describe algorithms based on upper-confidence-bounds applied to a restricted set of randomly selected arms and provide upper-bounds on the resulting expected regret. We also derive a lower-bound which matches (up to a logarithmic factor) the upper-bound in some cases.

## 1   Introduction

Multi-armed bandit problems describe typical situations where learning and optimization should be balanced in order to achieve good cumulative performances. Usual multi-armed bandit problems (see e.g. [8]) consider a finite number of possible actions (or arms) from which the learner may choose at each iteration. The number of arms is typically much smaller than the number of experiments allowed, so exploration of all possible options is usually performed and combined with exploitation of the apparently best ones.

In this paper, we investigate the case when the number of arms is infinite (or larger than the available number of experiments), which makes the exploration of all the arms an impossible task to achieve: if no additional assumption is made, it may be arbitrarily hard to find a near-optimal arm. Here we consider a stochastic assumption on the *mean-reward* of any new selected arm. When a new arm $k$ is pulled, its mean-reward $\mu_k$ is assumed to be an independent sample from a fixed distribution. Moreover, given the mean-reward $\mu_k$ for any arm $k$, the distribution of the *reward* is only required to be uniformly bounded and non-negative without any further assumption. Our assumptions essentially characterize the probability of pulling near-optimal arms. That is, given $\mu^* \in [0, 1]$ as the best possible mean-reward and $\beta \geq 0$ a parameter of the mean-reward distribution, the probability that a new arm is $\epsilon$-optimal is of order $\epsilon^\beta$ for small $\epsilon$, i.e. $\mathbb{P}(\mu_k \geq \mu^* - \epsilon) = \Theta(\epsilon^\beta)$ for $\epsilon \to 0$. Note that we write $f(\epsilon) = \Theta(g(\epsilon))$ for $\epsilon \to 0$ when $\exists c_1, c_2, \epsilon_0 > 0$ such that $\forall \epsilon \leq \epsilon_0, c_1 g(\epsilon) \leq f(\epsilon) \leq c_2 g(\epsilon)$.

---

[*]The major part of this work was completed during the research internship at Certis and INRIA SequeL.

Like in multi-armed bandits, this setting exhibits a trade off between exploitation (selection of the arms that are believed to perform well) and exploration. The exploration takes two forms here: discovery (pulling a new arm that has never been tried before) and sampling (pulling an arm already discovered in order to gain information about its actual mean-reward).

Numerous applications can be found e.g. in [5]. It includes labor markets (a worker has many opportunities for jobs), mining for valuable resources (such as gold or oil) when there are many areas available for exploration (the miner can move to another location or continue in the same location, depending on results), and path planning under uncertainty in which the path planner has to decide among a route that has proved to be efficient in the past (exploitation), or a known route that has not been explored many times (sampling), or a brand new route that has never been tried before (discovery).

Let us write $k_t$ the arm selected by our algorithm at time $t$. We define the regret up to time $n$ as $R_n = n\mu^* - \sum_{t=1}^{n} \mu_{k_t}$. From the tower rule, $\mathbb{E}R_n$ is the expectation of the difference between the rewards we would have obtained by drawing an optimal arm (an arm having a mean-reward equal to $\mu^*$) and the rewards we did obtain during the time steps $1, \ldots, n$. Our goal is to design an arm-pulling strategy such as to minimize this regret.

**Overview of our results:** We write $v_n = \tilde{O}(u_n)$ when for some $n_0, C > 0$, $v_n \leq C u_n (\log(u_n))^2$, for all $n \geq n_0$. We assume that the rewards of the arms lie in $[0, 1]$. Our regret bounds depend on whether $\mu^* = 1$ or $\mu^* < 1$. For $\mu^* = 1$, our algorithms are such that $\mathbb{E}R_n = \tilde{O}(n^{\beta/(1+\beta)})$. For $\mu^* < 1$, we have $\mathbb{E}R_n = \tilde{O}(n^{\beta/(1+\beta)})$ if $\beta > 1$, and (only) $\mathbb{E}R_n = \tilde{O}(n^{1/2})$ if $\beta \leq 1$. Moreover we derive the lower-bound: for any $\beta > 0$, $\mu^* \leq 1$, any algorithm satisfies $\mathbb{E}R_n \geq C n^{\beta/(1+\beta)}$ for some $C > 0$. Finally we propose an algorithm having the anytime property, which is based on an arm-increasing rule.

Our algorithms essentially consist in pulling $K$ different arms randomly chosen, where $K$ is of order $n^{\beta/2}$ if $\mu^* < 1$ and $\beta < 1$, and $n^{\beta/(1+\beta)}$ otherwise, and using a variant of the UCB (Upper Confidence Bound) algorithm ([3],[2]) on this set of $K$ arms, which takes into account the empirical variance of the rewards. This last point is crucial to get the proposed rate for $\mu^* = 1$ and $\beta < 1$, i.e. in cases where there are many arms with small variance.

**Previous works on many-armed bandits:** In [5], a specific setting of an infinitely many-armed bandit is considered, namely that the rewards are Bernoulli random variables with parameter $p$, where $p$ follows a uniform law over a given interval $[0, \mu^*]$. All mean-rewards are therefore in $[0, \mu^*]$. They proposed three algorithms. (1) The *1-failure strategy* where an arm is played as long as 1s are received. When a 0 is received, a new arm is played and this strategy is repeated forever. (2) The *m-run strategy* uses the 1-failure strategy until either $m$ continuous 1s are received (from the same arm) or $m$ different arms have been played. In the first case, we continue to play forever the current arm. In the second case, the arm that gave the most wins is chosen to play for the remaining rounds. Finally, (3) the *m-learning strategy* uses the 1-failure strategy during the first $m$ rounds, and for the remaining rounds it chooses the arm that gave the most 1s during the first $m$ rounds.

For $\mu^* = 1$, the authors of [5] have shown that 1-failure strategy, $\sqrt{n}$-run strategy, and $\log(n)\sqrt{n}$-learning strategy have a regret $\mathbb{E}R_n \leq 2\sqrt{n}$. They also provided a lower bound on the regret of any strategy: $\mathbb{E}R_n \geq \sqrt{2n}$. For $\mu^* < 1$, the corresponding optimal strategies are $\sqrt{n\mu^*}$-run strategy and $\sqrt{n\mu^*}\log(n\mu^*)$-learning strategy. All these algorithms require the knowledge of the horizon $n$ of the game. In many applications, it is important to design algorithms having the anytime property, that is, the upper bounds on the expected regret $\mathbb{E}R_n$ have the similar order for all $n$. Under the same Bernoulli assumption on the reward distributions, such algorithms has been obtained in [9].

In comparison to their setting (uniform distribution corresponds to $\beta = 1$), our upper- and lower-bounds are also of order $\sqrt{n}$ up to a logarithmic factor, and we do not assume that we know exactly the distribution of the mean-reward. However it is worth noting that the proposed algorithms in [5, 9] heavily depend on the Bernoulli assumption of the rewards and are not easily transposable to general distributions. Note also that the Bernoulli assumption does not work for the real problems mentioned above, where the outcomes may take several possible values.

Thus an important aspect of our work, compared to previous many-armed bandits, is that our setting allows general reward distributions for the arms, under a simple assumption on the mean-reward.

## 2 Main results

In our framework, each arm of a bandit is characterized by the distribution of the rewards (obtained by drawing that arm) and the essential parameter of the distribution of rewards is its expectation. Another parameter of interest is the standard deviation. With low variance, poor arms will be easier to spot while good arms will have higher probability of not being disregarded at the beginning due to unlucky trials. To draw an arm is equivalent to draw a distribution $\nu$ of mean-rewards. Let $\mu = \int w\nu(dw)$ and $\sigma^2 = \int (w-\mu)^2 \nu(dw)$ denote the expectation and variance of $\nu$. The quantities $\mu$ and $\sigma$ are random variables. Our assumptions are the following:

(A) Rewards are uniformly bounded: without loss of generality, we assume all rewards are in $[0,1]$.
(B) the expected reward of a randomly drawn arm satisfies: there exist $\mu^* \in (0,1]$ and $\beta > 0$ s.t.

$$\mathbb{P}\{\mu > \mu^* - \epsilon\} = \Theta(\epsilon^\beta), \text{ for } \epsilon \to 0 \tag{1}$$

(C) there is a function $V : [0,1] \to \mathbb{R}$ such that $\mathbb{P}\{\sigma^2 \leq V(\mu^* - \mu)\} = 1$.

The key assumption here is (B). It gives us (the order of) the number of arms that needs to be drawn before finding an arm that is $\epsilon$-close to the optimum[1] (i.e., an arm for which $\mu \geq \mu^* - \epsilon$). Assumption (B) implies that there exists positive constants $c_1$ and $c_2$ such that for any $\epsilon \in [0, \mu^*]$, we have[2]

$$c_1 \epsilon^\beta \leq \mathbb{P}\{\mu > \mu^* - \epsilon\} \leq \mathbb{P}\{\mu \geq \mu^* - \epsilon\} \leq c_2 \epsilon^\beta. \tag{2}$$

For example, the uniform distribution on $(0, \mu^*)$ satisfies the Condition (1) with $\beta = 1$.
Assumption (C) always holds for $V(u) = \mu^*(1 - \mu^* + u)$ (since $\mathbb{V}\text{ar}\, W \leq \mathbb{E}W(1 - \mathbb{E}W)$ when $W \in [0,1]$). However it is convenient when the near-optimal arms have low variance (for instance, this happens when $\mu^* = 1$).

Let $X_{k,1}, X_{k,2}, \ldots$ denote the rewards obtained when pulling arm $k$. These are i.i.d. random variables with common expected value denoted $\mu_k$. Let $\overline{X}_{k,s} \triangleq \frac{1}{s} \sum_{j=1}^s X_{k,j}$ and $V_{k,s} \triangleq \frac{1}{s} \sum_{j=1}^s (X_{k,j} - \overline{X}_{k,s})^2$ be the empirical mean and variance associated with the first $s$ draws of arm $k$. Let $T_k(t)$ denote the number of times arm $k$ is chosen by the policy during the first $t$ plays. We will use as a subroutine of our algorithms the following version of UCB (Upper Confidence Bound) algorithm as introduced in [2]. Let $(\mathcal{E}_t)_{t \geq 0}$ be a nondecreasing sequence of nonnegative real numbers. It will be referred to as the exploration sequence since the larger it is, the more UCB explores. For any arm $k$ and nonnegative integers $s, t$, introduce

$$B_{k,s,t} \triangleq \overline{X}_{k,s} + \sqrt{\frac{2V_{k,s}\mathcal{E}_t}{s}} + \frac{3\mathcal{E}_t}{s} \tag{3}$$

with the convention $1/0 = +\infty$. Define the UCB-V (for Variance estimate) policy:

> **UCB-V policy for a set $\mathcal{K}$ of arms:**
> At time $t$, play an arm in $\mathcal{K}$ maximizing $B_{k,T_k(t-1),t}$.

From [2, Theorem 1], the main property of $B_{k,s,t}$ is that with probability at least $1 - 5(\log t)e^{-\mathcal{E}_t/2}$, for any $s \in [0,t]$ we have $\mu_k \leq B_{k,s,t}$. So provided that $\mathcal{E}_t$ is large, $B_{k,T_k(t-1),t}$ is an observable quantity at time $t$ which upper bounds $\mu_k$ with high probability. We consider nondecreasing sequence $(\mathcal{E}_t)$ in order that these bounds hold with probability increasing with time. This ensures that the low probability event, that the algorithm might concentrate the draws on suboptimal arms, has a decreasing probability with time.

### 2.1 UCB revisited for the infinitely many-armed bandit

When the number of arms of the bandit is greater than the total number of plays, it makes no sense to apply UCB-V algorithm (or other variants of UCB [3]) since its first step is to draw each arm once (to have $B_{k,T_k(t-1),t}$ finite). A more meaningful and natural approach is to decide at the beginning

that only $K$ arms will be investigated in the entire experiment. The $K$ should be sufficiently small with respect to $n$ (the total number of plays), as in this way we have fewer plays on bad arms and most of the plays will be on the best of $K$ arms. The number $K$ should not be too small either, since we want that the best of the $K$ arms has an expected reward close to the best possible arm.

It is shown in [2, Theorem 4] that in the multi-armed bandit, taking a too small exploration sequence (e.g. such as $\mathcal{E}_t \leq \frac{1}{2}\log t$) might lead to polynomial regret (instead of logarithmic for e.g. $\mathcal{E}_t = 2\log t$) in a simple 2-armed bandit problem. However, we will show that this is not the case in the infinitely many-armed bandit, where one may (and should) take much smaller exploration sequences (typically of order $\log\log t$). The reason for this phenomenon is that in this setting, there are typically many near-optimal arms so that the subroutine UCB-V may miss some good arms (by unlucky trials) without being hurt: there are many other near-optimal arms to discover! This illustrates a trade off between the two aspects of exploration: sample the current, not well-known, arms or discover new arms.

We will start our analysis by considering the following UCB-V($\infty$) algorithm:

---

**UCB-V($\infty$) algorithm**: Given parameters $K$ and the exploration sequence $(\mathcal{E}_t)$
- Randomly choose $K$ arms,
- Run the UCB-V policy on the set of the $K$ selected arms.

---

**Theorem 1** *If the exploration sequence satisfies $2\log(10\log t) \leq \mathcal{E}_t \leq \log t$, then for $n \geq 2$ and $K \geq 2$ the expected regret of the UCB-V($\infty$) algorithm satisfies:*

$$\mathbb{E}R_n \leq C\left\{(\log K)nK^{-1/\beta} + K(\log n)\mathbb{E}\left[\left(\tfrac{V(\Delta)}{\Delta}+1\right)\wedge(n\Delta)\right]\right\}, \qquad (4)$$

*where $\Delta = \mu^* - \mu$ with $\mu$ the random variable corresponding to the expected reward of a sampled arm from the pool, and where $C$ is a positive constant depending only on $c_1$ and $\beta$ (see (2)).*

**Proof:** The UCB-V($\infty$) algorithm has two steps: randomly choose $K$ arms and run a UCB subroutine on the selected arms. The first part of the proof studies what happens during the UCB subroutine, that is, conditionally to the arms that have been randomly chosen during the first step of the algorithm. In particular we consider in the following that $\mu_1, \ldots, \mu_K$ are fixed. From the equality (obtained using Wald's theorem):

$$\mathbb{E}R_n = \sum_{k=1}^{K}\mathbb{E}\{T_k(n)\}\Delta_k \qquad (5)$$

with $\Delta_k = \mu^* - \mu_k$, it suffices to bound $\mathbb{E}T_k(n)$. The proof is inspired from the ones of Theorems 2 and 3 in [2]. The novelty of the following lemma is to include the product of probabilities in the last term of the right-hand-side. This enables us to incorporate the idea that if there are a lot of near-optimal arms, it is very unlikely that suboptimal arms are often drawn.

**Lemma 1** *For any real number $\tau$ and any positive integer $u$, we have*

$$\mathbb{E}T_k(n) \leq u + \sum_{t=u+1}^{n}\sum_{s=u}^{t}\mathbb{P}\big(B_{k,s,t} > \tau\big) + \sum_{t=u+1}^{n}\prod_{k'\neq k}\mathbb{P}(\exists s'\in[0,t],\ B_{k',s',t}\leq\tau)$$

$$(6)$$

*where the expectations and probabilities are conditioned on the set of selected arms.*

**Proof:** We have $T_k(n) - u \leq \sum_{t=u+1}^{n}Z_k(u,t)$ where $Z_k(u,t) = \mathbf{1}_{I_t=k;T_k(t)>u}$. We have

$$
\begin{aligned}
Z_k(u,t) &\leq \mathbf{1}_{\forall k'\neq k\ B_{k,T_k(t-1),t}\geq B_{k',T_{k'}(t-1),t};T_k(t-1)\geq u}\\
&\leq \mathbf{1}_{\exists s\in[u,t]\ B_{k,s,t}>\tau} + \mathbf{1}_{\forall k'\neq k\ \exists s'\in[0,t]\ B_{k',s',t}\leq\tau}
\end{aligned}
$$

where the last inequality holds since if the two terms in the last sum are equal to zero, then it implies that there exists $k' \neq k$ such that for any $s' \in [0,t]$ and any $s \in [u,t]$, $B_{k',s',t} > \tau \geq B_{k,s,t}$. Taking the expectation of both sides, using a union bound and the independence between rewards obtained from different arms, we obtain Lemma 1. $\square$

Now we use Inequality (6) with $\tau = \frac{\mu^*+\mu_k}{2} = \mu_k + \frac{\Delta_k}{2} = \mu^* - \frac{\Delta_k}{2}$, and $u$ the smallest integer larger than $32\left(\frac{\sigma_k^2}{\Delta_k^2} + \frac{1}{\Delta_k}\right)\log n$. These choices are made to ensure that the probabilities in the r.h.s.

of (6) are small. Precisely, for any $s \geq u$ and $t \leq n$, we have

$$\sqrt{\frac{2[\sigma_k^2 + \Delta_k/4]\mathcal{E}_t}{s}} + 3\frac{\mathcal{E}_t}{s} \leq \sqrt{\frac{[2\sigma_k^2 + \Delta_k/2]\log n}{u}} + 3\frac{\log n}{u}$$
$$\leq \sqrt{\frac{[2\sigma_k^2 + \Delta_k/2]\Delta_k^2}{32[\sigma_k^2 + \Delta_k]}} + \frac{3\Delta_k^2}{32[\sigma_k^2 + \Delta_k]} = \frac{\Delta_k}{4}\left[\sqrt{\frac{\sigma_k^2 + \Delta_k/4}{\sigma_k^2 + \Delta_k}} + \frac{3}{8}\frac{\Delta_k}{\sigma_k^2 + \Delta_k}\right] \leq \frac{\Delta_k}{4},$$

where the last inequality holds since it is equivalent to $(x - 1)^2 \geq 0$ for $x = \sqrt{\frac{\sigma_k^2 + \Delta_k/4}{\sigma_k^2 + \Delta_k}}$. Thus:

$$\mathbb{P}(B_{k,s,t} > \tau) \leq \mathbb{P}\left(\overline{X}_{k,s} + \sqrt{\frac{2V_{k,s}\mathcal{E}_t}{s}} + 3\frac{\mathcal{E}_t}{s} > \mu_k + \Delta_k/2\right)$$
$$\leq \mathbb{P}\left(\overline{X}_{k,s} + \sqrt{\frac{2[\sigma_k^2 + \Delta_k/4]\mathcal{E}_t}{s}} + 3\frac{\mathcal{E}_t}{s} > \mu_k + \Delta_k/2\right) + \mathbb{P}\left(V_{k,s} \geq \sigma_k^2 + \Delta_k/4\right)$$
$$\leq \mathbb{P}\left(\overline{X}_{k,s} - \mu_k > \Delta_k/4\right) + \mathbb{P}\left(\frac{\sum_{j=1}^{s}(X_{k,j} - \mu_k)^2}{s} - \sigma_k^2 \geq \Delta_k/4\right) \quad (7)$$
$$\leq 2e^{-s\Delta_k^2/(32\sigma_k^2 + 8\Delta_k/3)},$$

where in the last step we used Bernstein's inequality twice. Summing up we obtain

$$\sum_{s=u}^{t} \mathbb{P}(B_{k,s,t} > \tau) \leq 2\sum_{s=u}^{\infty} e^{-s\Delta_k^2/(32\sigma_k^2 + 8\Delta_k/3)} = 2\frac{e^{-u\Delta_k^2/(32\sigma_k^2 + 8\Delta_k/3)}}{1 - e^{-\Delta_k^2/(32\sigma_k^2 + 8\Delta_k/3)}}$$
$$\leq \left(\frac{80\sigma_k^2}{\Delta_k^2} + \frac{7}{\Delta_k}\right)e^{-u\Delta_k^2/(32\sigma_k^2 + 8\Delta_k/3)} \leq \left(\frac{80\sigma_k^2}{\Delta_k^2} + \frac{7}{\Delta_k}\right)n^{-1}, \quad (8)$$

where we have used that $1 - e^{-x} \geq 4x/5$ for $0 \leq x \leq 3/8$. Now let us bound the product of probabilities in (6). Since $\tau = \mu^* - \Delta_k/2$, we have

$$\prod_{k' \neq k} \mathbb{P}(\exists s \in [0, t], \ B_{k',s,t} \leq \tau) \leq \prod_{k':\mu_{k'} > \mu^* - \Delta_k/2} \mathbb{P}(\exists s \in [0, t], \ B_{k',s,t} < \mu_k').$$

Now from [2, Theorem 1], with probability at least $1 - 5(\log t)e^{-\mathcal{E}_t/2}$, for any $s \in [0, t]$ we have $\mu_k \leq B_{k,s,t}$. For $\mathcal{E}_t \geq 2\log(10\log t)$, this gives $\mathbb{P}(\exists s \in [0, t], \ B_{k',s,t} < \mu_k') \leq 1/2$. Putting all the bounds of the different terms of (6) leads to

$$\mathbb{E}T_k(n) \leq 1 + 32\left(\frac{\sigma_k^2}{\Delta_k^2} + \frac{1}{\Delta_k}\right)\log n + \left(\frac{80\sigma_k^2}{\Delta_k^2} + \frac{7}{\Delta_k}\right) + n2^{-N_{\Delta_k}},$$

with $N_{\Delta_k}$ the cardinal of $\{k' \in \{1, \ldots, K\} : \mu_{k'} > a - \Delta_k/2\}$. Since $\Delta_k \leq \mu^* \leq 1$ and $T_k(n) \leq n$, the previous inequality can be simplified into

$$\mathbb{E}T_k(n) \leq \left\{\left[50\left(\frac{\sigma_k^2}{\Delta_k^2} + \frac{1}{\Delta_k}\right)\log n\right] \wedge n\right\} + n2^{-N_{\Delta_k}}, \quad (9)$$

Here, for the sake of simplicity, we are not interested in having tight constants. From here on, we will take the expectations with respect to all sources of randomness, that is including the one coming from the first step of UCB-V($\infty$). The quantities $\Delta_1, \ldots, \Delta_K$ are i.i.d. random variables satisfying $0 \leq \Delta_k \leq \mu^*$ and $\mathbb{P}(\Delta_k \leq \epsilon) = \Theta(\epsilon^\beta)$. The quantities $\sigma_1, \ldots, \sigma_k$ are i.i.d. random variables satisfying almost surely $\sigma_k^2 \leq V(\Delta_k)$. From (5) and (9), we have

$$\mathbb{E}R_n = K\mathbb{E}\{T_1(n)\Delta_1\} \leq K\mathbb{E}\left\{\left[50\left(\frac{V(\Delta_1)}{\Delta_1} + 1\right)\log n\right] \wedge (n\Delta_1) + n\Delta_1 2^{-N_{\Delta_1}}\right\} \quad (10)$$

Let $p$ denote the probability that the expected reward $\mu$ of a randomly drawn arm satisfies $\mu > \mu^* - \delta/2$ for a given $\delta$. Conditioning on $\Delta_1 = \delta$, the quantity $N_{\Delta_1}$ follows a binomial distribution with parameters $K - 1$ and $p$, hence $\mathbb{E}(2^{-N_{\Delta_1}}|\Delta_1 = \delta) = (1 - p + p/2)^{K-1}$. By using (2), we get:

$$\mathbb{E}\{\Delta_1 2^{-N_{\Delta_1}}\} = \mathbb{E}\{\Delta_1 (1 - \mathbb{P}(\mu > \mu^* - \Delta_1/2)/2)^{K-1}\} \leq \mathbb{E}\chi(\Delta_1),$$

with $\chi(u) = u(1 - c_3 u^\beta)^{K-1}$ and $c_3 = c_1/2^\beta$. We have $\chi'(u) = (1 - c_3 u^\beta)^{K-2}[1 - c_3(1 + (K - 1)\beta)u^\beta]$ so that $\chi(u) \leq \chi(u_0)$ with $u_0 = \frac{1}{[c_3(1+(K-1)\beta)]^{1/\beta}}$ and $\chi(u_0) = \frac{(1 - \frac{1}{1+(K-1)\beta})^{K-1}}{[c_3(1+(K-1)\beta)]^{1/\beta}} \leq C'K^{-1/\beta}$ for $C'$ a positive constant depending only $c_1$ and $\beta$. For any $u_1 \in [u_0, \mu^*]$, we have

$$\mathbb{E}\chi(\Delta_1) \leq \chi(u_0)\mathbb{P}(\Delta_1 \leq u_1) + \chi(u_1)\mathbb{P}(\Delta_1 > u_1) \leq \chi(u_0)\mathbb{P}(\Delta_1 \leq u_1) + \chi(u_1).$$

Let us take $u_1 = C''\left(\frac{\log K}{K}\right)^{1/\beta}$ for $C''$ a positive constant depending on $c_1$ and $\beta$ sufficiently large to ensure $u_1 \geq u_0$ and $\chi(u_1) \leq K^{-1-1/\beta}$. We obtain $\mathbb{E}\chi(\Delta_1) \leq CK^{-1/\beta}\frac{\log K}{K}$ for an appropriate constant $C$ depending on $c_1$ and $\beta$. Putting this into (10), we obtain the result of Theorem 1. $\square$

The r.h.s. of Inequality (4) contains two terms. The first term is the bias: when we randomly draw $K$ arms, the expected reward of the best drawn arm is $\tilde{O}(K^{-1/\beta})$-optimal. So the best algorithm, once the $K$ arms are fixed, will yield a regret $\tilde{O}(nK^{-1/\beta})$. The second term is the estimation. It indicates the difference between the UCB subroutine's performance and the best drawn arm.

## 2.2 Strategy for fixed play number

Consider that we know in advance the total number of plays $n$ and the value of $\beta$. In this case, one can use the UCB-V($\infty$) algorithm with parameter $K$ of order of the minimizer of the r.h.s. of Inequality (4). This leads to the following UCB-F (for Fixed horizon) algorithm.

---

**UCB-F (fixed horizon)**: given total number of plays $n$, and parameters $\mu^*$ and $\beta$ of (1)
- Choose $K$ arms with $K$ of order $\begin{cases} n^{\frac{\beta}{2}} & \text{if } \beta < 1, \mu^* < 1 \\ n^{\frac{\beta}{\beta+1}} & \text{otherwise, i.e. if } \mu^* = 1 \text{ or } \beta \geq 1 \end{cases}$
- Run the UCB-V algorithm with the $K$ chosen arms and an exploration sequence satisfying

$$2\log(10\log t) \leq \mathcal{E}_t \leq \log t \qquad (11)$$

---

**Theorem 2** *For any $n \geq 2$, the expected regret of the UCB-F algorithm satisfies*

$$\mathbb{E}R_n \leq \begin{cases} C(\log n)\sqrt{n} & \text{if } \beta < 1 \text{ and } \mu^* < 1 \\ C(\log n)^2\sqrt{n} & \text{if } \beta = 1 \text{ and } \mu^* < 1 \\ C(\log n)n^{\frac{\beta}{1+\beta}} & \text{otherwise, i.e. if } \mu^* = 1 \text{ or } \beta > 1 \end{cases} \qquad (12)$$

*with $C$ a constant depending only on $c_1$, $c_2$ and $\beta$ (see (2)).*

**Proof:** The result comes from Theorem 1 by bounding the expectation $E = \mathbb{E}\left[\left(\frac{V(\Delta)}{\Delta}+1\right)\wedge(n\Delta)\right]$. First, as mentioned before, Assumption (C) is satisfied for $V(\Delta) = \mu^*(1-\mu^*+\Delta)$. So for $\mu^* = 1$ and this choice of function $V$, we have $E \leq 2$. For $\mu^* < 1$, since $\Delta \leq \mu^*$, we have $E \leq \mathbb{E}\Psi(\Delta)$ with $\Psi(t) = \frac{2\mu^*}{t} \wedge (nt)$. The function $\Psi$ is continuous and differentiable by parts. Using Fubini's theorem and Inequality (2), we have

$$\begin{aligned} \mathbb{E}\Psi(\Delta) &= \Psi(\mu^*) - \mathbb{E}\int_\Delta^{\mu^*}\Psi'(t)dt = \Psi(\mu^*) - \int_0^{\mu^*}\Psi'(t)\mathbb{P}(\Delta \leq t)dt \\ &\leq 2 + \int_{\sqrt{2/n}}^1 \frac{2}{t^2}c_2 t^\beta dt \leq \begin{cases} 2 + \frac{2^{(1+\beta)/2}c_2}{1-\beta}n^{\frac{1-\beta}{2}} & \text{if } \beta < 1 \\ 2 + c_2\log(n/2) & \text{if } \beta = 1 \\ 2 + \frac{2c_2}{\beta-1} & \text{if } \beta > 1 \end{cases}. \end{aligned}$$

Putting these bounds in Theorem 1, we get

$$\mathbb{E}R_n \leq \begin{cases} C\left\{(\log K)nK^{-1/\beta} + (\log n)Kn^{\frac{1-\beta}{2}}\right\} & \text{if } \beta < 1 \text{ and } \mu^* < 1 \\ C\left\{(\log K)nK^{-1/\beta} + (\log n)^2K\right\} & \text{if } \beta = 1 \text{ and } \mu^* < 1 \\ C\left\{(\log K)nK^{-1/\beta} + (\log n)K\right\} & \text{otherwise: } \mu^* = 1 \text{ or } \beta > 1 \end{cases}$$

with $C$ a constant only depending on $c_1$, $c_2$ and $\beta$. The number $K$ of selected arms in UCB-F is taken of the order of the minimizer of these bounds up to a logarithmic factor. $\square$

Theorem 2 makes no difference between a logarithmic exploration sequence and an iterated logarithmic exploration sequence. However in practice, it is clearly better to take an iterated logarithmic exploration sequence, for which the algorithm spends much less time on exploring all suboptimal arms. For sake of simplicity, we have fixed the constants in (11). It is easy to check that for $\mathcal{E}_t = \zeta \log_t$ and $\zeta \geq 1$, Inequality (12) still holds but with a constant $C$ depending linearly in $\zeta$.

Theorem 2 shows that when $\mu^* = 1$ or $\beta \geq 1$, the bandit subroutine takes no time in spotting near-optimal arms (the use of UCB-V algorithm using variance estimate is crucial for this), whereas for $\beta < 1$ and $\mu^* < 1$, which means a lot of near-optimal arms with possibly high variances, the bandit subroutine has difficulties in achieving low regret.

The next theorem shows that our regret upper bounds are optimal up to logarithmic terms except for the case $\beta < 1$ and $\mu^* < 1$. We do not know whether the rate $O(n^{\beta/2}\log n)$ for $\beta < 1$ and $\mu^* < 1$ is improvable. This remains an open problem.

**Theorem 3** *For any $\beta > 0$ and $\mu^* \leq 1$, any algorithm suffers a regret larger than $cn^{\frac{\beta}{1+\beta}}$ for some small enough constant $c$ depending on $c_2$ and $\beta$.*

**Sketch of proof.** If we want to have a regret smaller than $cn^{\beta/(1+\beta)}$ we need that most draws are done on an arm having an individual regret smaller than $\epsilon = cn^{-1/(1+\beta)}$. To find such an arm, we need to try a number of arms larger than $C'\epsilon^{-\beta} = C'c^{-\beta}n^{\beta/(1+\beta)}$ arms for some $C' > 0$ depending on $c_2$ and $\beta$. Since these arms are drawn at least once and since most of these arms give a constant regret, it leads to a regret larger than $C''c^{-\beta}n^{\beta/(1+\beta)}$ with $C''$ depending on $c_2$ and $\beta$. For $c$ small enough, this contradicts that the regret is smaller than $cn^{\beta/(1+\beta)}$. So it is not possible to improve on the $n^{\beta/(1+\beta)}$ rate. $\square$

## 2.3 Strategy for unknown play number

To apply the UCB-F algorithm we need to know the total number of plays $n$ and we choose the corresponding $K$ arms before starting. When $n$ is unknown ahead of time, we propose here an anytime algorithm with a simple and reasonable way of choosing $K$ by adding a new arm from time to time into the set of sampled arms. Let $K_n$ denote the number of arms played up to time $n$. We set $K_0 = 0$. We define the UCB-AIR (for Arm-Increasing Rule):

---

**UCB-AIR (Arm-Increasing Rule):** given parameters $\mu^*$ and $\beta$ of (1),
- At time $n$, try a new arm if

$$K_{n-1} < \begin{cases} n^{\frac{\beta}{2}} & \text{if } \beta < 1 \text{ and } \mu^* < 1 \\ n^{\frac{\beta}{\beta+1}} & \text{otherwise: } \mu^* = 1 \text{ or } \beta \geq 1 \end{cases}$$

- Otherwise apply UCB-V on $K_{n-1}$ drawn arms with an exploration sequence satisfying

$$2\log(10\log t) \leq \mathcal{E}_t \leq \log t$$

---

This arm-increasing rule makes our algorithm applicable for the anytime problem. This is a more reasonable approach in practice than restarting-based algorithms like the ones using the doubling trick (see e.g. [4, Section 5.3]). Our second main result is to show that the UCB-AIR algorithm has the same properties as the UCB-F algorithm (proof omitted from this extended abstract).

**Theorem 4** *For any horizon time $n \geq 2$, the expected regret of the UCB-AIR algorithm satisfies*

$$\mathbb{E}R_n \leq \begin{cases} C(\log n)^2\sqrt{n} & \text{if } \beta < 1 \text{ and } \mu^* < 1 \\ C(\log n)^2 n^{\frac{\beta}{1+\beta}} & \text{otherwise, i.e. if } \mu^* = 1 \text{ or } \beta \geq 1 \end{cases} \tag{13}$$

*with $C$ a constant depending only on $c_1$, $c_2$ and $\beta$ (see (2)).*

## 3 Comparison with continuum-armed bandits and conclusion

In continuum-armed bandits (see e.g. [1, 6, 4]), an infinity of arms is also considered. The arms lie in some Euclidean (or metric) space and their mean-reward is a deterministic and smooth (e.g. Lipschitz) function of the arms. This setting is different from ours since our assumption is stochastic and does not consider regularities of the mean-reward w.r.t. the arms. However, if we choose an arm-pulling strategy which consists in selecting randomly the arms, then our setting encompasses continuum-armed bandits. For example, consider the domain $[0, 1]^d$ and a mean-reward function $\mu$ assumed to be locally equivalent to a Hölder function (of order $\alpha \in [0, +\infty)$) around any maximum $x^*$ (the number of maxima is assumed to be finite), i.e.

$$\mu(x^*) - \mu(x) = \Theta(\|x^* - x\|^\alpha) \text{ when } x \to x^*. \tag{14}$$

Pulling randomly an arm $X$ according to the Lebesgue measure on $[0, 1]^d$, we have: $\mathbb{P}(\mu(X) > \mu^* - \epsilon) = \Theta(\mathbb{P}(\|X - x^*\|^\alpha < \epsilon)) = \Theta(\epsilon^{d/\alpha})$, for $\epsilon \to 0$. Thus our assumption (1) holds with $\beta = d/\alpha$, and our results say that if $\mu^* = 1$, we have $\mathbb{E}R_n = \tilde{O}(n^{\beta/(1+\beta)}) = \tilde{O}(n^{d/(\alpha+d)})$.

For $d = 1$, under the assumption that $\mu$ is $\alpha$-Hölder (i.e. $|\mu(x) - \mu(y)| \leq c\|x - y\|^\alpha$ for $0 < \alpha \leq 1$), [6] provides upper- and lower-bounds on the regret $R_n = \Theta(n^{(\alpha+1)/(2\alpha+1)})$. Our results gives

$\mathbb{E}R_n = \tilde{O}(n^{1/(\alpha+1)})$ which is better for all values of $\alpha$. The reason for this apparent contradiction is that the lower bound in [6] is obtained by the construction of a very irregular function, which actually does not satisfy our local assumption (14).

Now, under assumptions (14) for any $\alpha > 0$ (around a finite set of maxima), [4] provides the rate $\mathbb{E}R_n = \tilde{O}(\sqrt{n})$. Our result gives the same rate when $\mu^* < 1$ but in the case $\mu^* = 1$ we obtain the improved rate $\mathbb{E}R_n = \tilde{O}(n^{1/(\alpha+1)})$ which is better whenever $\alpha > 1$ (because we are able to exploit the low variance of the good arms). Note that like our algorithm, the algorithms in [4] as well as in [6], do not make an explicit use (in the procedure) of the smoothness of the function. They just use a 'uniform' discretization of the domain.

On the other hand, the zooming algorithm of [7] adapts to the smoothness of $\mu$ (more arms are sampled at areas where $\mu$ is high). For any dimension $d$, they obtain $\mathbb{E}R_n = \tilde{O}(n^{(d'+1)/(d'+2)})$, where $d' \leq d$ is their 'zooming dimension'. Under assumptions (14) we deduce $d' = \frac{\alpha-1}{\alpha}d$ using the Euclidean distance as metric, thus their regret is $\mathbb{E}R_n = \tilde{O}(n^{(d(\alpha-1)+\alpha)/(d(\alpha-1)+2\alpha)})$. For locally quadratic functions (i.e. $\alpha = 2$), their rate is $\tilde{O}(n^{(d+2)/(d+4)})$, whereas ours is $\tilde{O}(n^{d/(2+d)})$. Again, we have a smaller regret although we do not use the smoothness of $\mu$ in our algorithm. Here the reason is that the zooming algorithm does not make full use of the fact that the function is locally quadratic (it considers a Lipschitz property only). However, in the case $\alpha < 1$, our rates are worse than algorithms specifically designed for continuum armed bandits.

Hence, the comparison between the many-armed and continuum-armed bandits settings is not easy because of the difference in nature of the basis assumptions. Our setting is an alternative to the continuum-armed bandit setting which does not require the existence of an underlying metric space in which the mean-reward function would be smooth. Our assumption (1) naturally deals with possibly very complicated functions where maxima may be located in any part of the space. For the continuum-armed bandit problems when there are relatively many near-optimal arms, our algorithm will be also competitive compared to the specifically designed continuum-armed bandit algorithms. This result matches the intuition that in such cases, a random selection strategy will perform well.

To conclude, our contributions are: (i) Compared to previous results on many-armed bandits, our setting allows general mean-reward distributions for the arms, under a simple assumption on the probability of pulling near-optimal arms. (ii) We show that, for infinitely many-armed bandits, we need much less exploration of each arm than for finite-armed bandits (the $\log$ term may be replaced by $\log \log$). (iii) Our variant of UCB algorithm, making use of the variance estimate, enables to obtain higher rates in cases when the variance of the near-optimal arms is small. (iv) We propose the UCB-AIR algorithm, which is anytime, taking advantage of an arm-increasing rule. (v) We provide a lower-bound matching the upper-bound (up to a logarithmic factor) in the case $\beta \geq 1$ or $\mu^* = 1$.

## Footnotes

[1] Precise computations lead to a number which is of order $\epsilon^{-\beta}$ up to possibly a logarithmic factor.

[2] Indeed, (1) implies that for some $0 < c_1' < c_2'$, there exists $0 < \epsilon_0 < \mu^*$ such that for any $\epsilon \leq \epsilon_0$, $c_1' \epsilon^\beta \leq \mathbb{P}\{\mu > \mu^* - \epsilon\} \leq \mathbb{P}\{\mu \geq \mu^* - \epsilon\} \leq c_2' \epsilon^\beta$. Then one may take $c_1 = c_1' \epsilon_0^\beta$ and $c_2 = \max(\epsilon_0^{-\beta}, c_2')$.

## References

[1] R. Agrawal. The continuum-armed bandit problem. *SIAM J. Control and Optimization*, 33:1926–1951, 1995.

[2] J.-Y. Audibert, R. Munos, and C. Szepesvári. Tuning bandit algorithms in stochastic environments. In M. Hutter, R. A. Servedio, and E. Takimoto, editors, *ALT*, volume 4754 of *Lecture Notes in Computer Science*, pages 150–165. Springer, 2007.

[3] P. Auer, N. Cesa-Bianchi, and P. Fischer. Finite-time analysis of the multiarmed bandit problem. *Machine Learning*, 47(2/3):235–256, 2002.

[4] P. Auer, R. Ortner, and C. Szepesvári. Improved rates for the stochastic continuum-armed bandit problem. *20th COLT, San Diego, CA, USA*, 2007.

[5] D. A. Berry, R. W. Chen, A. Zame, D. C. Heath, and L. A. Shepp. Bandit problems with infinitely many arms. *The Annals of Statistics*, 25(5):2103–2116, 1997.

[6] R. Kleinberg. Nearly tight bounds for the continuum-armed bandit problem. In *NIPS-2004*, 2004.

[7] R. Kleinberg, A. Slivkins, and E. Upfal. Multi-armed bandit problems in metric spaces. In *Proceedings of the 40th ACM Symposium on Theory of Computing*, 2008.

[8] T. L. Lai and H. Robbins. Asymptotically efficient adaptive allocation rules. *Advances in Applied Mathematics*, 6:4–22, 1985.

[9] O. Teytaud, S. Gelly, and M. Sebag. Anytime many-armed bandit. *Conférence francophone sur l'Apprentissage automatique (CAp) Grenoble, France*, 2007.

